# Connectionist Models for Auditory Scene Analysis

Richard O. Duda
Department of Electrical Engineering
San Jose State University
San Jose, CA 95192

## Abstract

Although the visual and auditory systems share the same basic tasks of informing an organism about its environment, most connectionist work on hearing to date has been devoted to the very different problem of speech recognition. We believe that the most fundamental task of the auditory system is the analysis of acoustic signals into components corresponding to individual sound sources, which Bregman has called auditory scene analysis. Computational and connectionist work on auditory scene analysis is reviewed, and the outline of a general model that includes these approaches is described.

## 1   INTRODUCTION

The primary task of any perceptual system is to tell us about the external world. The primary problem is that the sensory inputs provide too much data and too little information. A perceptual system must glean from the flood of incomplete, noisy, redundant and constantly changing streams of data those invariant properties that reveal important objects and events in the environment. For humans, the perceptual systems with the widest bandwidths are the visual system and the auditory system. There are many obvious similarities and differences between these modalities, and in addition to using them to perceive different aspects of the physical world, we also use them in quite different ways to communicate with one another.

The earliest neural-network models for vision and hearing addressed problems in pattern recognition, with optical character recognition and isolated word recognition among the first engineering applications. However, about twenty years ago the research goals in vision and hearing began to diverge. In particular, the need for computers to perceive the external environment motivated vision researchers to seek the principles and procedures for recovering information about the physical world from visual data (Marr, 1982; Ballard and Brown, 1982). By contrast, the vast majority of work on machine audition remained focused on the communication problem of speech recognition (Morgan and Scofield, 1991; Rabiner and Juang, 1993). While this focus has produced considerable progress, the resulting systems are still not very robust, and perform poorly in uncontrolled environments. Furthermore, as Richards (1988) has noted, "... Speech, like writing and reading, is a specialized skill of advanced animals, and understanding speech need not be the best route to understanding how we interpret the patterns of natural sounds that comprise most of the acoustic spectrum about us."

In recent years, some researchers concerned with modeling audition have begun to shift their attention from speech understanding to sound understanding. The inspiration for much of this activity has come from the work of Bregman, whose book on auditory scene analysis documents experimental evidence for important gestalt principles that summarize the ways that people group elementary events in frequency/time into sound objects or streams (Bregman, 1990). In this survey paper, we briefly review this activity and consider its implications for the development of connectionist models for auditory scene analysis.

## 2   AUDITORY SCENE ANALYSIS

In vision, Marr (1982) emphasized the importance of identifying the tasks of the visual system and developing a computational theory that is distinct from particular algorithms or implementations. The computational theory had to specify the problems to be solved, the sensory data that is available, and the additional knowledge or assumptions required to solve the problems. Among the various tasks of the visual system, Marr believed that the recovery of the three-dimensional shapes of the surfaces of objects from the sensory image data was the most fundamental.

The auditory system also has basic tasks that are more primitive than the recognition of speech. These include (1) the separation of different sound sources, (2) the localization of the sources in space (3) the suppression of echoes and reverberation, (4) the decoupling of sources from the environment, (5) the characterization of the sources, and (6) the characterization of the environment. Unfortunately, the relation between physical sound sources and perceived sound streams is not a simple one-to-one correspondence. Distributed sound sources, echoes, and synthetic sounds can easily confuse auditory perception. Nevertheless, humans still do much better at these six basic tasks than any machine hearing system that exists today.

From the standpoint of physics, the raw data available for performing these tasks is the pair of acoustic signals arriving at the two ears. From the standpoint of neurophysiology, the raw data is the activity on the auditory nerve. The nonlinear, mechano-neural spectral analysis performed by the cochlea converts sound pressure fluctuations into auditory nerve firings. For better or for worse, the cochlea

decomposes the signal into many frequency components, transforming it into a frequency/time (or, more accurately, a place/time) spectrogram-like representation. The auditory system must find the underlying order in this dynamic flow of data.

For a specific case, consider a simple musical mixture of several periodic signals. Within its limits of resolution, the cochlea decomposes each individual signal into its discrete harmonic components. Yet, under ordinary circumstances, we do not hear these components as separate sounds, but rather we fuse them into a single sound having, as musicians say, its particular timbre or tone color. However, if there is something distinctive about the different signals (such as different pitch or different modulation), we do not fuse all of the sounds together, but rather hear the separate sources, each with its own timbre.

What information is available to group the spectral components into sound streams? Hartmann (1988) identifies the following factors that influence grouping: (1) common onset/offset, (2) common harmonic relations, (3) common modulation, (4) common spatial origin, (5) continuity of spectral envelope, (6) duration, (7) sound pressure level, and (8) context. These properties are easier to name than to precisely specify, and it is not surprising that no current model incorporates them all. However, several auditory scene analysis systems have been built that exploit some subset of these cues (Weintraub, 1985; Cooke, 1993; Mellinger, 1991; Brown, 1992; Brown and Cooke, 1993; Ellis, 1993). Although these are computational rather than connectionist models, most of them at least find inspiration in the structure of the mammalian auditory system.

# 3  NEURAL AND CONNECTIONIST MODELS

The neural pathways from the cochlea through the brainstem nuclei to the auditory cortex are complex, but have been extensively investigated. Although this system is far from completely understood, neurons in the brainstem nuclei are known to be sensitive to various acoustic features — onsets, offsets and modulation in the dorsal cochlear nucleus, interaural time differences (ITD's) in the medial superior olive (MSO), interaural intensity differences (IID's) in the lateral superior olive (LSO), and spatial location maps in the inferior colliculus (Pickles, 1988).

Both functional and connectionist models have been developed for all of these functions. Because it is both important and relatively well understood, the cochlea has received by far the most attention (Allen, 1985). As a result of this work, we now have real-time implementations for some of these models as analog VLSI chips (Lyon and Mead, 1988; Lazzaro et al., 1993). Connectionist models for sound localization have also been extensively explored. Indeed, one of the earliest of all neural network models was Jeffress's classic crosscorrelation model (Jeffress, 1948), which was hypothesized forty years before neural crosscorrelation structures were actually found in the barn owl (Carr and Konishi, 1988). Models have subsequently been proposed for both the LSO (Reed and Blum, 1990) and the MSO (Han and Colburn, 1991). Mathematically, both the ITD and IID cues for binaural localization are exposed by crosscorrelation. Lyon showed that crosscorrelation can also be used to separate as well as localize the signals (Lyon, 1983). VLSI crosscorrelation chips can provide this information in real time (Lazzaro and Mead, 1989; Bhadkamkar and Fowler, 1993).

While interaural crosscorrelation can determine the azimuth to a sound source, full three-dimensional localization also requires the determination of elevation and range. Because of a lack of symmetry in the orientation of its ears, the barn owl can actually determine azimuth from the ITD and elevation from the IID. This at least in part explains why it has been such a popular choice for connectionist modeling (Spence et al., 1990; Moiseff et al., 1991; Palmieri et al., 1991; Rosen, Rumelhart and Knudsen, 1993). Unfortunately, the localization mechanisms used by humans are more complicated.

It is well known that humans use monaural, spectral shape cues to estimate elevation in the median sagittal plane (Blauert, 1983; Middlebrooks and Green, 1991), and source localization models based on this approach have been developed (Neti, Young and Schneider, 1992; Zakarauskas and Cynander, 1993). The author has shown that there are strong binaural cues for elevation at short distances away from the median plane, and has used statistical methods to estimate both azimuth and elevation accurately from IID data alone (Duda, 1994). In addition, backprop models have been developed that can estimate azimuth and elevation from IID and ITD inputs jointly (Backman and Karjalainen 93; Anderson, Gilkey and Janko, 1994).

Finally, psychologists have long been aware of an important reverberation-suppression phenomenon known as the precedence effect or the law of the first wavefront (Zurek, 1987). It is usually summarized by saying that echoes of a sound source have little effect on its localization, and are not even consciously heard if they are not delayed more than the so-called echo threshold, which ranges from 5-10 ms for sharp clicks to more than 50 ms for music. It is generally believed that the precedence effect can be accounted for by contralateral inhibition in the crosscorrelation process, and Lindemann has accounted for many of the phenomena by a conceptually simple connectionist model (Lindemann, 1986).

However, Clifton and her colleagues have found that the echoes are indeed heard if the timing of the echoes suddenly changes, as might happen when one moves from one acoustic environment into another one (Clifton 1987; Freyman, Clifton and Litovsky, 1991). Clifton conjectures that the auditory system is continually analyzing echo patterns to model the acoustic environment, and that the resulting expectations modify the echo threshold. This suggests that simple crosscorrelation models will not be adequate when the listener is moving, and thus that even the localization problem is still unsolved.

## 4   ARCHITECTURE OF AN AUDITORY MODEL

If we look back at the six basic tasks for the auditory system, we see that only one (source localization) has received much attention from connectionist researchers, and its solution is incomplete. In particular, current localization models cannot handle multiple sources and cannot cope with significant room echoes and reverberation. The common problem for all of the basic tasks is that of source separation, which only the auditory scene analysis systems have addressed.

Fig. 1 shows a functional block diagram for a hypothetical auditory model that combines the computational and connectionist models and has the potential of coping with a multisource environment. The inputs to the model are the left-ear

and right-ear signals, and the main output is a dynamic set of streams. The system is primarily data driven, although low-bandwidth efferent paths could be added for tasks such as automatic gain control.

Data flow on the left half of the diagram is monaural, and dataflow of the right half is binaural. The binaural processing is based on crosscorrelation analysis of the cochlear outputs. The author has shown that interaural differences not only effective in determining azimuth, but can also be used to determine elevation as well (Duda, 1994). We have chosen to follow Slaney and Lyon (Slaney and Lyon, 1993) in basing the monaural analysis on autocorrelation analysis. Originally proposed by Licklider (1951) to explain pitch phenomena, autocorrelation is a biologically plausible operation that supports the common onset, modulation and harmonicity analysis needed for stream formation (Duda, Lyon and Slaney, 1990; Brown and Cooke, 1993).

While the processes at lower levels of this diagram are relatively well understood, the process of stream formation is problematic. Bregman (1990) has posed this problem in terms of grouping the components of the "neural spectrogram" in both frequency and time. He has identified two principles that seem to be employed in stream formation: exclusive allocation (a component may not be used in more than one description at a time) and accounting (all incoming components must be assigned to some source). The various auditory scene analysis systems that we mentioned earlier provide different mechanisms for exploiting these principles to form auditory streams. Unfortunately, the principles admit of exceptions, and the existing implementations seem rather ad hoc and arbitrary. The development of a biologically plausible model for stream formation is the central unsolved problem for connectionist research in audition.

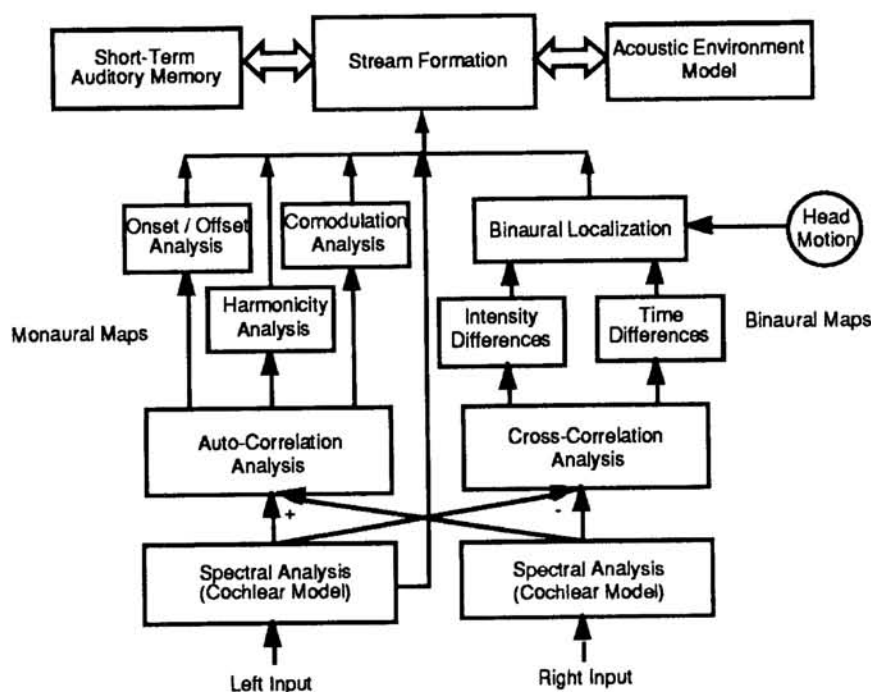

Figure 1: Block diagram for a basic auditory model

## Acknowledgements

This work was supported by the National Science Foundation under NSF Grant No. IRI-9214233. This paper could not have been written without the many discussions on these topics with Al Bregman, Dick Lyon, David Mellinger, Bernard MontReynaud, John R. Pierce, Malcolm Slaney and J. Martin Tenenbaum, and from the stimulating CCRMA Hearing Seminar at Stanford University that Bernard initiated and that Malcolm has maintained and invigorated.

## References

Allen, J. B. (1985). "Cochlear modeling," *IEEE ASSP Magazine*, vol. 2, pp. 3-29.

Anderson, T. R., R. H. Gilkey and J. A. Janko (1994). "Using neural networks to model human sound localization," in T. Anderson and R. H. Gilkey (eds.), *Binaural and Spatial Hearing*. Hillsdale, NJ: Lawrence Erlbaum Associates.

Backman, J. and M. Karjalainen (1993). "Modelling of human directional and spatial hearing using neural networks," *ICASSP93*, pp. I-125-I-128. (Minneapolis, MN).

Bhadkamkar, N. and B. Fowler (1993). "A sound localization system based on biological analogy," *1993 IEEE International Conference on Neural Networks*, pp. 1902-1907. (San Francisco, CA).

Ballard, D. H. and C. M. Brown (1982). *Computer Vision*. Englewood Cliffs, NJ: Prentice-Hall.

Blauert, J. P. (1983). *Spatial Hearing*. Cambridge, MA: MIT Press.

Bregman, A. S. (1990). *Auditory Scene Analysis*. Cambridge, MA: MIT Press, 1990.

Brown, G. J. (1992). "Computational auditory scene analysis: A representational approach," PhD dissertation, Department of Computer Science, University of Sheffield, Sheffield, England, UK.

Brown, G. J. and M. Cooke (1993). "Physiologically-motivated signal representations for computational auditory scene analysis," in M. Cooke, S. Beet and M. Crawford (eds.), *Visual Representations of Speech Signals*, pp. 181-188. Chichester, England: John Wiley and Sons.

Carr, C. E. and M. Konishi (1988). "Axonal delay lines for time measurement in the owl's brainstem," *Proc. Nat. Acad. Sci. USA*, vol. 85, pp. 8311-8315.

Clifton, R. K. (1987). "Breakdown of echo suppression in the precedence effect," *J. Acoust. Soc. Am.*, vol. 82, pp. 1834-1835.

Cooke, M. P. (1993). *Modelling Auditory Processing and Organisation*. Cambridge, UK: Cambridge University Press.

Duda, R. O., R. F. Lyon and M. Slaney (1990). "Correlograms and the separation of sounds," *Proc. 24th Asilomar Conf. on Signals, Systems and Computers*, pp. 457-461 (Asilomar, CA).

Duda, R. O. (1994). "Elevation dependence of the interaural transfer function," in T. Anderson and R. H. Gilkey (eds.), *Binaural and Spatial Hearing.* Hillsdale, NJ: Lawrence Erlbaum Associates.

Ellis, D. P. W. (1993). "Hierarchic models of hearing for sound separation and reconstruction," *1993 IEEE Workshop on Applications of Signal Processing to Audio and Acoustics.*

Freyman, R. L., R. K. Clifton and R. Y. Litovsky (1991). "Dynamic processes in the precedence effect," *J. Acoust. Soc. Am.*, vol. 90, pp. 874-884.

Han, Y. and H. S. Colburn (1991). "A neural cell model of MSO," *Proc. 1991 IEEE Seventeenth Annual Northeast Bioenginering Conference*, pp. 121-122 (Hartford, CT).

Hartmann, W. A. (1988). "Pitch perception and the segregation and integration of auditory entities," in G. M. Edelman, W. E. Gail and W. M. Cowan (eds.), *Auditory Function.* New York, NY: John Wiley and Sons, Inc.

Jeffress, L. A. (1948). "A place theory of sound localization," *J. Comp. Physiol. Psychol.*, vol. 41, pp. 35-39.

Lazzaro, J. and C. A. Mead (1989). "A silicon model of auditory localization," *Neural Computation*, vol. 1, pp. 47-57.

Lazzaro, J., J. Wawrzynek, M. Mahowald, M. Sivilotti and D. Gillespie (1993). "Silicon auditory processors as computer peripherals," *IEEE Transactions on Neural Networks*, vol. 4, pp. 523-528.

Licklider, J. C. R. (1951). "A duplex theory of pitch perception," *Experentia*, vol. 7, pp. 128-133.

Lindemann, W. (1986). "Extension of a binaural cross-correlation model by contralateral inhibition. I. Simulation of lateralization for stationary signals," *J. Acoust. Soc. Am.*, vol. 80, pp. 1608-1622; II. The law of the first wave front," *J. Acoust. Soc. Am.*, vol. 80, pp. 1623-1630.

Lyon, R. F. (1983). "A computational model of binaural localization and separation," *ICASSP83*, pp. 1148-1151. (Boston, MA).

Lyon, R. F. and C. Mead (1988). "An analog electronic cochlea," *IEEE Trans. Acoustics, Speech and Signal Processing*, vol. 36, pp. 1119-1134.

Marr, D. (1982). *Vision.* San Francisco, CA: W. H. Freeman and Company.

Mellinger, D. K. (1991). "Event formation and separation of musical sound," PhD dissertation, Department of Music, Stanford University, Stanford, CA; Report No. STAN-M-77, Center for Computer Research in Music and Acoustics, Stanford University, Stanford, CA.

Middlebrooks, J. C. and D. M. Green (1991). "Sound localization by human listeners," *Annu. Rev. Psychol.*, vol. 42, pp. 135-159.

Moiseff, A. et al. (1991). "An artificial neural network for studying binaural sound localization," *Proc. 1991 IEEE Seventeenth Annual Northeast Bioengineering Conference*, pp. 1-2 (Hartford, CT).

Morgan, D. P. and C. L. Scofield (1991). *Neural Networks and Speech Processing.* Boston, MA: Kluwer Academic Publishers.

Neti, C., E. D. Young and M. H. Schneider (1992). "Neural network models of sound localization based on directional filtering by the pinna," *J. Acoust. Soc. Am.,* vol. 92, pp. 3140-3156.

Palmieri, F., M. Datum, A. Shah and A. Moiseff (1991). "Sound localization with a neural network trained with the multiple extended Kalman algorithm," *Proc. Int. Joint Conf. on Neural Networks*, pp. I125-I131 (Seattle, WA).

Pickles, James O. (1988). *An Introduction to the Physiology of Hearing, 2nd edition.* London, Academic Press, 1988.

Rabiner, L. and B-H Juang (1993). *Fundamentals of Speech Recognition.* Englewood Cliffs, NJ: Prentice-Hall.

Reed, M. C. and J. J. Blum (1990). "A model for the computation and encoding of azimuthal information by the lateral superior olive," *J. Acoust. Soc. Am.,* vol. 88, pp. 1442-1453.

Richards, W. (1988). "Sound interpretation," in W. Richards (ed.), *Natural Computation*, pp. 301-308. Cambridge, MA: MIT Press.

Rosen, D. , D. Rumelhart and E. Knudsen (1993). "A connectionist model of the owl's localization system," in J. D. Cowan, G. Tesauro and J. Alspector (eds.), *Advances in Neural Information Processing Systems 6*. San Francisco, CA: Morgan Kaufmann Publishers.

Slaney, M. and R. F. Lyon (1993). "On the importance of time — A temporal representation of sound," in M. Cooke, S. Beet and M. Crawford (eds.), *Visual Representations of Speech Signals*, pp. 95-116. Chichester, England: John Wiley and Sons.

Spence, C. D. and J. C. Pearson (1990). "The computation of sound source elevation in the barn owl," in D. S. Touretzsky (ed.), *Advances in Neural Information Processing Systems 2*, pp. 10-17. San Mateo, CA: Morgan Kaufmann.

Weintraub, M. (1985). "A theory and computational model of auditory monaural sound separation," PhD dissertation, Department of Electrical Engineering, Stanford University, Stanford, CA.

Zakarauskas, P. and M. S. Cynander (1993). "A computational theory of spectral cue localization," *J. Acoust. Soc. Am.,* vol. 94, pp. 1323-1331.

Zurek, P. M. (1987). "The precedence effect," in W. A. Yost and G. Gourevitch (eds.) *Directional Hearing*, pp. 85-106. New York, NY: Springer Verlag.
